# AUC Optimization vs. Error Rate Minimization

**Corinna Cortes**[*] **and Mehryar Mohri**
AT&T Labs – Research
180 Park Avenue, Florham Park, NJ 07932, USA
{corinna, mohri}@research.att.com

## Abstract

The *area under an ROC curve* (AUC) is a criterion used in many applications to measure the quality of a classification algorithm. However, the objective function optimized in most of these algorithms is the error rate and not the AUC value. We give a detailed statistical analysis of the relationship between the AUC and the error rate, including the first exact expression of the expected value and the variance of the AUC for a fixed error rate. Our results show that the average AUC is monotonically increasing as a function of the classification accuracy, but that the standard deviation for uneven distributions and higher error rates is noticeable. Thus, algorithms designed to minimize the error rate may not lead to the best possible AUC values. We show that, under certain conditions, the global function optimized by the RankBoost algorithm is exactly the AUC. We report the results of our experiments with RankBoost in several datasets demonstrating the benefits of an algorithm specifically designed to globally optimize the AUC over other existing algorithms optimizing an approximation of the AUC or only locally optimizing the AUC.

## 1  Motivation

In many applications, the overall classification error rate is not the most pertinent performance measure, criteria such as *ordering* or *ranking* seem more appropriate. Consider for example the list of relevant documents returned by a search engine for a specific query. That list may contain several thousand documents, but, in practice, only the top fifty or so are examined by the user. Thus, a search engine's ranking of the documents is more critical than the accuracy of its classification of all documents as relevant or not. More generally, for a binary classifier assigning a real-valued score to each object, a better correlation between output scores and the probability of correct classification is highly desirable.

A natural criterion or summary statistic often used to measure the ranking quality of a classifier is the *area under an ROC curve* (AUC) [8].[1] However, the objective function optimized by most classification algorithms is the error rate and not the AUC. Recently, several algorithms have been proposed for maximizing the AUC value locally [4] or maximizing some approximations of the global AUC value [9, 15], but, in general, these algorithms do not obtain AUC values significantly better than those obtained by an algorithm designed to minimize the error rates. Thus, it is important to determine the relationship between the AUC values and the error rate.

---

[*]This author's new address is: Google Labs, 1440 Broadway, New York, NY 10018, corinna@google.com.

[1]The AUC value is equivalent to the Wilcoxon-Mann-Whitney statistic [8] and closely related to the Gini index [1]. It has been re-invented under the name of L-measure by [11], as already pointed out by [2], and slightly modified under the name of Linear Ranking by [13, 14].

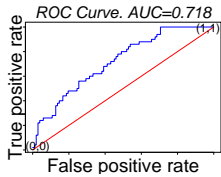

$$\text{True positive rate} = \frac{\text{correctly classified positive}}{\text{total positive}}$$

$$\text{False positive rate} = \frac{\text{incorrectly classified negative}}{\text{total negative}}$$

Figure 1: An example of ROC curve. The line connecting $(0,0)$ and $(1,1)$, corresponding to random classification, is drawn for reference. The true positive (negative) rate is sometimes referred to as the *sensitivity* (resp. *specificity*) in this context.

In the following sections, we give a detailed statistical analysis of the relationship between the AUC and the error rate, including the first exact expression of the expected value and the variance of the AUC for a fixed error rate.[2] We show that, under certain conditions, the global function optimized by the RankBoost algorithm is exactly the AUC. We report the results of our experiments with RankBoost in several datasets and demonstrate the benefits of an algorithm specifically designed to globally optimize the AUC over other existing algorithms optimizing an approximation of the AUC or only locally optimizing the AUC.

## 2 Definition and properties of the AUC

The *Receiver Operating Characteristics* (ROC) curves were originally developed in signal detection theory [3] in connection with radio signals, and have been used since then in many other applications, in particular for medical decision-making. Over the last few years, they have found increased interest in the machine learning and data mining communities for model evaluation and selection [12, 10, 4, 9, 15, 2].

The ROC curve for a binary classification problem plots the true positive rate as a function of the false positive rate. The points of the curve are obtained by sweeping the classification threshold from the most positive classification value to the most negative. For a fully random classification, the ROC curve is a straight line connecting the origin to $(1,1)$. Any improvement over random classification results in an ROC curve at least partially above this straight line. Fig. (1) shows an example of ROC curve. The AUC is defined as the area under the ROC curve and is closely related to the ranking quality of the classification as shown more formally by Lemma 1 below.

Consider a binary classification task with $m$ positive examples and $n$ negative examples. We will assume that a classifier outputs a strictly ordered list for these examples and will denote by $1_X$ the indicator function of a set $X$.

**Lemma 1 ([8])** *Let $c$ be a fixed classifier. Let $x_1, \ldots, x_m$ be the output of $c$ on the positive examples and $y_1, \ldots, y_n$ its output on the negative examples. Then, the AUC, $A$, associated to $c$ is given by:*

$$A = \frac{\sum_{i=1}^{m} \sum_{j=1}^{n} 1_{x_i > y_j}}{mn} \tag{1}$$

*that is the value of the* Wilcoxon-Mann-Whitney statistic *[8].*

*Proof.* The proof is based on the observation that the AUC value is exactly the probability $P(X > Y)$ where $X$ is the random variable corresponding to the distribution of the outputs for the positive examples and $Y$ the one corresponding to the negative examples [7]. The Wilcoxon-Mann-Whitney statistic is clearly the expression of that probability in the discrete case, which proves the lemma [8]. □

Thus, the AUC can be viewed as a measure based on pairwise comparisons between classifications of the two classes. With a perfect ranking, all positive examples are ranked higher than the negative ones and $A = 1$. Any deviation from this ranking decreases the AUC.

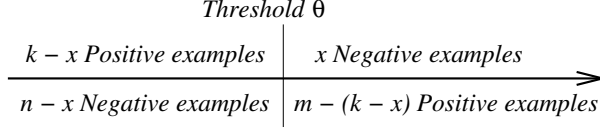

*Threshold* θ

| $k - x$ Positive examples | $x$ Negative examples |
| --- | --- |
| $n - x$ Negative examples | $m - (k - x)$ Positive examples |

Figure 2: For a fixed number of errors $k$, there may be $x, 0 \le x \le k$, false negative examples.

## 3   The Expected Value of the AUC

In this section, we compute *exactly* the expected value of the AUC over all classifications with a fixed number of errors and compare that to the error rate.

Different classifiers may have the same error rate but different AUC values. Indeed, for a given classification threshold $\theta$, an arbitrary reordering of the examples with outputs more than $\theta$ clearly does not affect the error rate but leads to different AUC values. Similarly, one may reorder the examples with output less than $\theta$ without changing the error rate.

Assume that the number of errors $k$ is fixed. We wish to compute the average value of the AUC over all classifications with $k$ errors. Our model is based on the simple assumption that all classifications or rankings with $k$ errors are equiprobable. One could perhaps argue that errors are not necessarily evenly distributed, e.g., examples with very high or very low ranks are less likely to be errors, but we cannot justify such biases in general.

For a given classification, there may be $x, 0 \le x \le k$, false positive examples. Since the number of errors is fixed, there are $k - x$ false negative examples. Figure 3 shows the corresponding configuration. The two regions of examples with classification outputs above and below the threshold are separated by a vertical line. For a given $x$, the computation of the AUC, $A$, as given by Eq. (1) can be divided into the following three parts:

$$A = \frac{A_1 + A_2 + A_3}{mn}, \qquad \text{with} \tag{2}$$

$A_1 = $ the sum over all pairs $(x_i, y_j)$ with $x_i$ and $y_j$ in distinct regions;
$A_2 = $ the sum over all pairs $(x_i, y_j)$ with $x_i$ and $y_j$ in the region above the threshold;
$A_3 = $ the sum over all pairs $(x_i, y_j)$ with $x_i$ and $y_j$ in the region below the threshold.

The first term, $A_1$, is easy to compute. Since there are $(m - (k - x))$ positive examples above the threshold and $n - x$ negative examples below the threshold, $A_1$ is given by:

$$A_1 = (m - (k - x))(n - x) \tag{3}$$

To compute $A_2$, we can assign to each negative example above the threshold a position based on its classification rank. Let position one be the first position above the threshold and let $\alpha_1 < \ldots < \alpha_x$ denote the positions in increasing order of the $x$ negative examples in the region above the threshold. The total number of examples classified as positive is $N = m - (k - x) + x$. Thus, by definition of $A_2$,

$$A_2 = \sum_{i=1}^{x} (N - \alpha_i) - (x - i) \tag{4}$$

where the first term $N - \alpha_i$ represents the number of examples ranked higher than the $i$th example and the second term $x - i$ discounts the number of negative examples incorrectly ranked higher than the $i$th example. Similarly, let $\alpha'_1 < \ldots < \alpha'_{k-x}$ denote the positions of the $k - x$ positive examples below the threshold, counting positions in reverse by starting from the threshold. Then, $A_3$ is given by:

$$A_3 = \sum_{j=1}^{x'} (N' - \alpha'_j) - (x' - j) \tag{5}$$

with $N' = n - x + (k - x)$ and $x' = k - x$. Combining the expressions of $A_1$, $A_2$, and $A_3$ leads to:

$$A = \frac{A_1 + A_2 + A_3}{mn} = 1 + \frac{(k - 2x)^2 + k}{2mn} - \frac{\left( \sum_{i=1}^{x} \alpha_i + \sum_{j=1}^{x'} \alpha'_j \right)}{mn} \tag{6}$$

**Lemma 2** *For a fixed $x$, the average value of the AUC $A$ is given by:*

$$< A >_x = 1 - \frac{\frac{x}{n} + \frac{k-x}{m}}{2} \tag{7}$$

*Proof.* The proof is based on the computation of the average values of $\sum_{i=1}^{x} \alpha_i$ and $\sum_{j=1}^{x'} \alpha'_j$ for a given $x$. We start by computing the average value $< \alpha_i >_x$ for a given $i$, $1 \le i \le x$. Consider all the possible positions for $\alpha_1 \ldots \alpha_{i-1}$ and $\alpha_{i+1} \ldots \alpha_x$, when the value of $\alpha_i$ is fixed at say $\alpha_i = l$. We have $i \le l \le N - (x - i)$ since there need to be at least $i - 1$ positions before $\alpha_i$ and $N - (x - i)$ above. There are $l - 1$ possible positions for $\alpha_1 \ldots \alpha_{i-1}$ and $N - l$ possible positions for $\alpha_{i+1} \ldots \alpha_x$. Since the total number of ways of choosing the $x$ positions for $\alpha_1 \ldots \alpha_x$ out of $N$ is $\binom{N}{x}$, the average value $< \alpha_i >_x$ is:

$$< \alpha_i >_x = \frac{\sum_{l=i}^{N-(x-i)} l \binom{l-1}{i-1} \binom{N-l}{x-i}}{\binom{N}{x}} \tag{8}$$

Thus,

$$< \sum_{i=1}^{x} \alpha_i >_x = \frac{\sum_{i=1}^{x} \sum_{l=i}^{N-(x-i)} l \binom{l-1}{i-1} \binom{N-l}{x-i}}{\binom{N}{x}} = \frac{\sum_{l=1}^{N} l \sum_{i=1}^{x} \binom{l-1}{i-1} \binom{N-l}{x-i}}{\binom{N}{x}} \tag{9}$$

Using the classical identity: $\sum_{p_1+p_2=p} \binom{u}{p_1} \binom{v}{p_2} = \binom{u+v}{p}$, we can write:

$$< \sum_{i=1}^{x} \alpha_i >_x = \frac{\sum_{l=1}^{N} l \binom{N-1}{x-1}}{\binom{N}{x}} = \frac{N(N+1)}{2} \frac{\binom{N-1}{x-1}}{\binom{N}{x}} = \frac{x(N+1)}{2} \tag{10}$$

Similarly, we have:

$$< \sum_{j=1}^{x'} \alpha'_j >_x = \frac{x'(N'+1)}{2} \tag{11}$$

Replacing $< \sum_{i=1}^{x} \alpha_i >_x$ and $< \sum_{j=1}^{x'} \alpha'_j >_x$ in Eq. (6) by the expressions given by Eq. (10) and Eq. (11) leads to:

$$< A >_x = 1 + \frac{(k-2x)^2 + k - x(N+1) - x'(N'+1)}{2mn} = 1 - \frac{\frac{x}{n} + \frac{k-x}{m}}{2} \tag{12}$$

which ends the proof of the lemma. $\qquad\square$

Note that Eq. (7) shows that the average AUC value for a given $x$ is simply one minus the average of the accuracy rates for the positive and negative classes.

**Proposition 1** *Assume that a binary classification task with $m$ positive examples and $n$ negative examples is given. Then, the expected value of the AUC $A$ over all classifications with $k$ errors is given by:*

$$< A > = 1 - \frac{k}{m+n} - \frac{(n-m)^2(m+n+1)}{4mn} \left( \frac{k}{m+n} - \frac{\sum_{x=0}^{k-1} \binom{m+n}{x}}{\sum_{x=0}^{k} \binom{m+n+1}{x}} \right) \tag{13}$$

*Proof.* Lemma 2 gives the average value of the AUC for a fixed value of $x$. To compute the average over all possible values of $x$, we need to weight the expression of Eq. (7) with the total number of possible classifications for a given $x$. There are $\binom{N}{x}$ possible ways of choosing the positions of the $x$ misclassified negative examples, and similarly $\binom{N'}{x'}$ possible ways of choosing the positions of the $x' = k - x$ misclassified positive examples. Thus, in view of Lemma 2, the average AUC is given by:

$$< A > = \frac{\sum_{x=0}^{k} \binom{N}{x} \binom{N'}{x'} (1 - \frac{\frac{x}{n} + \frac{k-x}{m}}{2})}{\sum_{x=0}^{k} \binom{N}{x} \binom{N'}{x'}} \tag{14}$$

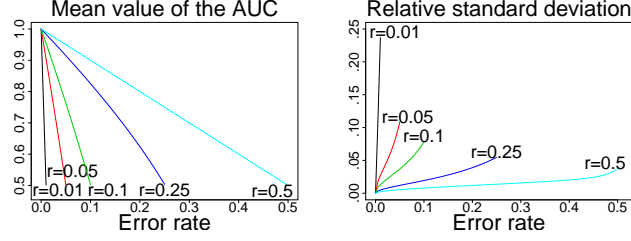

Figure 3: Mean (left) and relative standard deviation (right) of the AUC as a function of the error rate. Each curve corresponds to a fixed ratio of $r = n/(n+m)$. The average AUC value monotonically increases with the accuracy. For $n = m$, as for the top curve in the left plot, the average AUC coincides with the accuracy. The standard deviation decreases with the accuracy, and the lowest curve corresponds to $n = m$.

This expression can be simplified into Eq. (13)[3] using the following novel identities:

$$\sum_{x=0}^{k} \binom{N}{x}\binom{N'}{x'} = \sum_{x=0}^{k} \binom{n+m+1}{x} \tag{15}$$

$$\sum_{x=0}^{k} x \binom{N}{x}\binom{N'}{x'} = \sum_{x=0}^{k} \frac{(k-x)(m-n)+k}{2} \binom{n+m+1}{x} \tag{16}$$

that we obtained by using Zeilberger's algorithm[4] and numerous combinatorial 'tricks'. $\square$

From the expression of Eq. (13), it is clear that the average AUC value is identical to the accuracy of the classifier only for even distributions ($n = m$). For $n \neq m$, the expected value of the AUC is a monotonic function of the accuracy, see Fig. (3)(left). For a fixed ratio of $n/(n+m)$, the curves are obtained by increasing the accuracy from $n/(n+m)$ to 1. The average AUC varies monotonically in the range of accuracy between $0.5$ and $1.0$. In other words, on average, there seems nothing to be gained in designing specific learning algorithms for maximizing the AUC: a classification algorithm minimizing the error rate also optimizes the AUC. However, this only holds for the average AUC. Indeed, we will show in the next section that the variance of the AUC value is not null for any ratio $n/(n+m)$ when $k \neq 0$.

## 4   The Variance of the AUC

Let $D = mn + \frac{(k-2x)^2+k}{2}$, $a = \sum_{i=1}^{x} \alpha_i$, $a' = \sum_{j=1}^{x'} \alpha'_j$, and $\alpha = a + a'$. Then, by Eq. (6), $mnA = D - \alpha$. Thus, the variance of the AUC, $\sigma^2(A)$, is given by:

$$\begin{aligned} (mn)^2 \sigma^2(A) &= \; <(D-\alpha)^2 - (<D> - <\alpha>)^2> \tag{17} \\ &= \; <D^2> - <D>^2 + <\alpha^2> - <\alpha>^2 - 2(<\alpha D> - <\alpha><D>) \end{aligned}$$

As before, to compute the average of a term $X$ over all classifications, we can first determine its average $<X>_x$ for a fixed $x$, and then use the function $F$ defined by:

$$F(Y) = \frac{\sum_{x=0}^{k} \binom{N}{x}\binom{N'}{x'} Y}{\sum_{x=0}^{k} \binom{N}{x}\binom{N'}{x'}} \tag{18}$$

and $<X> = F(<X>_x)$. A crucial step in computing the exact value of the variance of the AUC is to determine the value of the terms of the type $<a^2>_x = <(\sum_{i=1}^{x} \alpha_i)^2>_x$.

**Lemma 3** *For a fixed $x$, the average of $(\sum_{i=1}^{x} \alpha_i)^2$ is given by:*

$$< a^2 >_x = \frac{x(N+1)}{12}(3Nx + 2x + N) \tag{19}$$

*Proof.* By definition of $a$, $< a^2 >_x = b + 2c$ with:

$$b = < \sum_{i=1}^{x} \alpha_i^2 >_x \qquad c = < \sum_{1 \le i < j \le x} \alpha_i \alpha_j >_x \tag{20}$$

Reasoning as in the proof of Lemma 2, we can obtain:

$$b = \frac{\sum_{i=1}^{x} \sum_{l=i}^{N-(x-i)} l^2 \binom{l-1}{i-1}\binom{N-l}{x-i}}{\binom{N}{x}} = \sum_{l=1}^{N} l^2 \frac{\binom{N-1}{x-1}}{\binom{N}{x}} = \frac{(N+1)(2N+1)x}{6} \tag{21}$$

To compute $c$, we start by computing the average value of $< \alpha_i \alpha_j >_x$, for a given pair $(i, j)$ with $i < j$. As in the proof of Lemma 2, consider all the possible positions of $\alpha_1 \ldots \alpha_{i-1}$, $\alpha_{i+1} \ldots \alpha_{j-1}$, and $\alpha_{j+1} \ldots \alpha_x$ when $\alpha_i$ is fixed at $\alpha_i = l$, and $\alpha_j$ is fixed at $\alpha_j = l'$. There are $l - 1$ possible positions for the $\alpha_1 \ldots \alpha_{i-1}$, $l' - l - 1$ possible positions for $\alpha_{i+1} \ldots \alpha_{j-1}$, and $N - l'$ possible positions for $\alpha_{j+1} \ldots \alpha_x$. Thus, we have:

$$< \alpha_i \alpha_j >_x = \frac{\sum_{i \le l < l' \le N - (x-j)} ll' \binom{l-1}{i-1}\binom{l'-l-1}{j-i-1}\binom{N-l'}{x-j}}{\binom{N}{x}} \tag{22}$$

and

$$c = \frac{\sum_{l<l'} ll' \sum_{m_1+m_2+m_3=x-2} \binom{l-1}{m_1}\binom{l'-l-1}{m_2}\binom{N-l'}{m_3}}{\binom{N}{x}} \tag{23}$$

Using the identity $\sum_{m_1+m_2+m_3=x-2} \binom{l-1}{m_1}\binom{l'-l-1}{m_2}\binom{N-l'}{m_3} = \binom{N-2}{x-2}$, we obtain:

$$c = \frac{(N+1)(3N+2)x(x-1)}{24} \tag{24}$$

Combining Eq. (21) and Eq. (24) leads to Eq. (19). $\square$

**Proposition 2** *Assume that a binary classification task with $m$ positive examples and $n$ negative examples is given. Then, the variance of the AUC $A$ over all classifications with $k$ errors is given by:*

$$\sigma^2(A) = F((1 - \frac{\frac{x}{n} + \frac{k-x}{m}}{2})^2) - F((1 - \frac{\frac{x}{n} + \frac{k-x}{m}}{2}))^2 + \tag{25}$$

$$F(\frac{mx^2 + n(k-x)^2 + (m(m+1)x + n(n+1)(k-x)) - 2x(k-x)(m+n+1)}{12m^2n^2})$$

*Proof.* Eq. (18) can be developed and expressed in terms of $F$, $D$, $a$, and $a'$:

$$(mn)^2 \sigma^2(A) = F([D - < a + a' >_x]^2) - F(D - < a + a' >_x)^2 +$$
$$F(< a^2 >_x - < a >_x^2) + F(< a'^2 >_x - < a' >_x^2) \tag{26}$$

The expressions for $< a >_x$ and $< a' >_x$ were given in the proof of Lemma 2, and that of $< a^2 >_x$ by Lemma 3. The following formula can be obtained in a similar way: $< a'^2 >_x = \frac{x'(N'+1)}{12}(3N'x' + 2x' + N')$. Replacing these expressions in Eq. (26) and further simplifications give exactly Eq. (25) and prove the proposition. $\square$

The expression of the variance is illustrated by Fig. (3)(right) which shows the value of one standard deviation of the AUC divided by the corresponding mean value of the AUC. This figure is parallel to the one showing the mean of the AUC (Fig. (3)(left)). Each line is obtained by fixing the ratio $n/(n+m)$ and varying the number of errors from 1 to the size of the smallest class. The more uneven class distributions have the highest variance, the variance increases with the number of errors. These observations contradict the inexact claim of [15] that the variance is zero for all error rates with even distributions $n = m$. In Fig. (3)(right), the even distribution $n = m$ corresponds to the lowest dashed line.

| Dataset | Size | # of Attr. | $\frac{n}{n+m}$ (%) | AUCsplit[4] | | RankBoost | |
|---|---|---|---|---|---|---|---|
| | | | | Accuracy (%) | AUC (%) | Accuracy (%) | AUC (%) |
| Breast-Wpbc | 194 | 33 | 23.7 | $69.5 \pm 10.6$ | $59.3 \pm 16.2$ | $65.5 \pm 13.8$ | $80.4 \pm 8.0$ |
| Credit | 653 | 15 | 45.3 | | | $81.0 \pm 7.4$ | $94.5 \pm 2.9$ |
| Ionosphere | 351 | 34 | 35.9 | $89.6 \pm 5.0$ | $89.7 \pm 6.7$ | $83.6 \pm 10.9$ | $98.0 \pm 3.3$ |
| Pima | 768 | 8 | 34.9 | $72.5 \pm 5.1$ | $76.7 \pm 6.0$ | $69.7 \pm 7.6$ | $84.8 \pm 6.5$ |
| SPECTF | 269 | 43 | 20.4 | | | 67.3 | 93.4 |
| Page-blocks | 5473 | 10 | 10.2 | $96.8 \pm 0.2$ | $95.1 \pm 6.9$ | $92.0 \pm 2.5$ | $98.5 \pm 1.5$ |
| Yeast (CYT) | 1484 | 8 | 31.2 | $71.1 \pm 3.6$ | $73.3 \pm 4.0$ | $45.3 \pm 3.8$ | $78.5 \pm 3.0$ |

Table 1: Accuracy and AUC values for several datasets from the UC Irvine repository. The values for RankBoost are obtained by 10-fold cross-validation. The values for AUCsplit are from [4].

## 5  Experimental Results

Proposition 2 above demonstrates that, for uneven distributions, classifiers with the same fixed (low) accuracy exhibit noticeably different AUC values. This motivates the use of algorithms directly optimizing the AUC rather than doing so indirectly via minimizing the error rate. Under certain conditions, RankBoost [5] can be viewed exactly as an algorithm optimizing the AUC. In this section, we make the connection between RankBoost and AUC optimization, and compare the performance of RankBoost to two recent algorithms proposed for optimizing an approximation [15] or locally optimizing the AUC [4].

The objective of RankBoost is to produce a ranking that minimizes the number of incorrectly ordered pairs of examples, possibly with different costs assigned to the mis-rankings. When the examples to be ranked are simply two disjoint sets, the objective function minimized by RankBoost is

$$\text{rloss} = \sum_{i=1}^{m} \sum_{j=1}^{n} \frac{1}{m} \frac{1}{n} 1_{x_i \leq y_j} \tag{27}$$

which is exactly one minus the Wilcoxon-Mann-Whitney statistic. Thus, by Lemma 1, the objective function maximized by RankBoost coincides with the AUC.

RankBoost's optimization is based on combining a number of weak rankings. For our experiments, we chose as weak rankings threshold rankers with the range $\{0, 1\}$, similar to the boosted stumps often used by AdaBoost [6]. We used the so-called *Third Method* of RankBoost for selecting the best weak ranker. According to this method, at each step, the weak threshold ranker is selected so as to maximize the AUC of the weighted distribution. Thus, with this method, the global objective of obtaining the best AUC is obtained by selecting the weak ranking with the best AUC at each step.

Furthermore, the RankBoost algorithm maintains a perfect 50-50% distribution of the weights on the positive and negative examples. By Proposition 1, for even distributions, the mean of the AUC is identical to the classification accuracy. For threshold rankers like step functions, or stumps, there is no variance of the AUC, so the mean of the AUC is equal to the observed AUC. That is, instead of viewing RankBoost as selecting the weak ranker with the best weighted AUC value, one can view it as selecting the weak ranker with the lowest weighted error rate. This is similar to the choice of the best weak learner for boosted stumps in AdaBoost. So, for stumps, AdaBoost and RankBoost differ only in the updating scheme of the weights: RankBoost updates the positive examples differently from the negative ones, while AdaBoost uses one common scheme for the two groups.

Our experimental results corroborate the observation that RankBoost is an algorithm optimizing the AUC. RankBoost based on boosted stumps obtains AUC values that are substantially better than those reported in the literature for algorithms designed to locally or approximately optimize the AUC. Table 1 compares the results of RankBoost on a number of datasets from the UC Irvine repository to the results reported by [4]. The results for RankBoost are obtained by 10-fold cross-validation. For RankBoost, the accuracy and the best AUC values reported on each line of the table correspond to the same boosting step.

RankBoost consistently outperforms AUCsplit in a comparison based on AUC values, even for the datasets such as Breast-Wpbc and Pima where the two algorithms obtain similar accuracies. The table also lists results for the UC Irvine Credit Approval and SPECTF heart dataset, for which the authors of [15] report results corresponding to their AUC optimization algorithms. The AUC values reported by [15] are no better than 92.5% for the Credit

Approval dataset and only $87.5\%$ for the SPECTF dataset, which is substantially lower. From the table, it is also clear that RankBoost is not an error rate minimization algorithm. The accuracy for the Yeast (CYT) dataset is as low as $45\%$.

## 6  Conclusion

A statistical analysis of the relationship between the AUC value and the error rate was given, including the first exact expression of the expected value and standard deviation of the AUC for a fixed error rate. The results offer a better understanding of the effect on the AUC value of algorithms designed for error rate minimization. For uneven distributions and relatively high error rates, the standard deviation of the AUC suggests that algorithms designed to optimize the AUC value may lead to substantially better AUC values. Our experimental results using RankBoost corroborate this claim.

In separate experiments we have observed that AdaBoost achieves significantly better error rates than RankBoost (as expected) but that it also leads to AUC values close to those achieved by RankBoost. It is a topic for further study to explain and understand this property of AdaBoost. A partial explanation could be that, just like RankBoost, AdaBoost maintains at each boosting round an equal distribution of the weights for positive and negative examples.

## Footnotes

[2] An attempt in that direction was made by [15], but, unfortunately, the authors' analysis and the result are both wrong.

[3]An essential difference between Eq. (14) and the expression given by [15] is the weighting by the number of configurations. The authors' analysis leads them to the conclusion that the average AUC is identical to the accuracy for all ratios $n/(n+m)$, which is false.

[4]We thank Neil Sloane for having pointed us to Zeilberger's algorithm and Maple package.

## References

[1] L. Breiman, J. H. Friedman, R. A. Olshen, and C. J. Stone. *Classification and Regression Trees*. Wadsworth International, Belmont, CA, 1984.

[2] J-H. Chauchat, R. Rakotomalala, M. Carloz, and C. Pelletier. Targeting customer groups using gain and cost matrix; a marketing application. Technical report, ERIC Laboratory - University of Lyon 2, 2001.

[3] J. P. Egan. *Signal Detection Theory and ROC Analysis*. Academic Press, 1975.

[4] C. Ferri, P. Flach, and J. Hernández-Orallo. Learning decision trees using the area under the ROC curve. In *ICML-2002*. Morgan Kaufmann, 2002.

[5] Y. Freund, R. Iyer, R. E. Schapire, and Y. Singer. An efficient boosting algorithm for combining preferences. In *ICML-98*. Morgan Kaufmann, San Francisco, US, 1998.

[6] Yoav Freund and Robert E. Schapire. A decision theoretical generalization of on-line learning and an application to boosting. In *Proceedings of the Second European Conference on Computational Learning Theory*, volume 2, 1995.

[7] D. M. Green and J. A Swets. *Signal detection theory and psychophysics*. New York: Wiley, 1966.

[8] J. A. Hanley and B. J. McNeil. The meaning and use of the area under a receiver operating characteristic (ROC) curve. *Radiology*, 1982.

[9] M. C. Mozer, R. Dodier, M. D. Colagrosso, C. Guerra-Salcedo, and R. Wolniewicz. Prodding the ROC curve. In *NIPS-2002*. MIT Press, 2002.

[10] C. Perlich, F. Provost, and J. Simonoff. Tree induction vs. logistic regression: A learning curve analysis. *Journal of Machine Learning Research*, 2003.

[11] G. Piatetsky-Shapiro and S. Steingold. Measuring lift quality in database marketing. In *SIGKDD Explorations*. ACM SIGKDD, 2000.

[12] F. Provost and T. Fawcett. Analysis and visualization of classifier performance: Comparison under imprecise class and cost distribution. In *KDD-97*. AAAI, 1997.

[13] S. Rosset. Ranking-methods for flexible evaluation and efficient comparison of 2-class models. Master's thesis, Tel-Aviv University, 1999.

[14] S. Rosset, E. Neumann, U. Eick, N. Vatnik, and I. Idan. Evaluation of prediction models for marketing campaigns. In *KDD-2001*. ACM Press, 2001.

[15] L. Yan, R. Dodier, M. C. Mozer, and R. Wolniewicz. Optimizing Classifier Performance Via the Wilcoxon-Mann-Whitney Statistics. In *ICML-2003*, 2003.
